# Double Q-learning

**Hado van Hasselt**
Multi-agent and Adaptive Computation Group
Centrum Wiskunde & Informatica

## Abstract

In some stochastic environments the well-known reinforcement learning algorithm Q-learning performs very poorly. This poor performance is caused by large overestimations of action values. These overestimations result from a positive bias that is introduced because Q-learning uses the maximum action value as an approximation for the maximum expected action value. We introduce an alternative way to approximate the maximum expected value for any set of random variables. The obtained double estimator method is shown to sometimes underestimate rather than overestimate the maximum expected value. We apply the double estimator to Q-learning to construct Double Q-learning, a new off-policy reinforcement learning algorithm. We show the new algorithm converges to the optimal policy and that it performs well in some settings in which Q-learning performs poorly due to its overestimation.

## 1   Introduction

Q-learning is a popular reinforcement learning algorithm that was proposed by Watkins [1] and can be used to optimally solve Markov Decision Processes (MDPs) [2]. We show that Q-learning's performance can be poor in stochastic MDPs because of large overestimations of the action values. We discuss why this occurs and propose an algorithm called Double Q-learning to avoid this overestimation. The update of Q-learning is

$$Q_{t+1}(s_t, a_t) = Q_t(s_t, a_t) + \alpha_t(s_t, a_t) \left( r_t + \gamma \max_a Q_t(s_{t+1}, a) - Q_t(s_t, a_t) \right) \ . \tag{1}$$

In this equation, $Q_t(s, a)$ gives the value of the action $a$ in state $s$ at time $t$. The reward $r_t$ is drawn from a fixed reward distribution $R : S \times A \times S \to \mathbb{R}$, where $E\{r_t | (s, a, s') = (s_t, a_t, s_{t+1})\} = R_{sa}^{s'}$. The next state $s_{t+1}$ is determined by a fixed state transition distribution $P : S \times A \times S \to [0, 1]$, where $P_{sa}^{s'}$ gives the probability of ending up in state $s'$ after performing $a$ in $s$, and $\sum_{s'} P_{sa}^{s'} = 1$. The learning rate $\alpha_t(s, a) \in [0, 1]$ ensures that the update averages over possible randomness in the rewards and transitions in order to converge in the limit to the optimal action value function. This optimal value function is the solution to the following set of equations [3]:

$$\forall s, a : Q^*(s, a) = \sum_{s'} P_{sa}^{s'} \left( R_{sa}^{s'} + \gamma \max_a Q^*(s', a) \right) \ . \tag{2}$$

The discount factor $\gamma \in [0, 1)$ has two interpretations. First, it can be seen as a property of the problem that is to be solved, weighing immediate rewards more heavily than later rewards. Second, in non-episodic tasks, the discount factor makes sure that every action value is finite and therefore well-defined. It has been proven that Q-learning reaches the optimal value function $Q^*$ with probability one in the limit under some mild conditions on the learning rates and exploration policy [4–6].

Q-learning has been used to find solutions on many problems [7–9] and was an inspiration to similar algorithms, such as Delayed Q-learning [10], Phased Q-learning [11] and Fitted Q-iteration [12], to name some. These variations have mostly been proposed in order to speed up convergence rates

compared to the original algorithm. The convergence rate of Q-learning can be exponential in the number of experiences [13], although this is dependent on the learning rates and with a proper choice of learning rates convergence in polynomial time can be obtained [14]. The variants named above can also claim polynomial time convergence.

**Contributions**  An important aspect of the Q-learning algorithm has been overlooked in previous work: the use of the max operator to determine the value of the next state can cause large over-estimations of the action values. We show that Q-learning can suffer a large performance penalty because of a positive bias that results from using the maximum value as approximation for the maximum expected value. We propose an alternative double estimator method to find an estimate for the maximum value of a set of stochastic values and we show that this sometimes underestimates rather than overestimates the maximum expected value. We use this to construct the new Double Q-learning algorithm.

The paper is organized as follows. In the second section, we analyze two methods to approximate the maximum expected value of a set of random variables. In Section 3 we present the Double Q-learning algorithm that extends our analysis in Section 2 and avoids overestimations. The new algorithm is proven to converge to the optimal solution in the limit. In Section 4 we show the results on some experiments to compare these algorithms. Some general discussion is presented in Section 5 and Section 6 concludes the paper with some pointers to future work.

## 2   Estimating the Maximum Expected Value

In this section, we analyze two methods to find an approximation for the maximum expected value of a set of random variables. The single estimator method uses the maximum of a set of estimators as an approximation. This approach to approximate the value of the maximum expected value is positively biased, as discussed in previous work in economics [15] and decision making [16]. It is a bias related to the Winner's Curse in auctions [17, 18] and it can be shown to follow from Jensen's inequality [19]. The double estimator method uses two estimates for each variable and uncouples the selection of an estimator and its value. We are unaware of previous work that discusses it. We analyze this method and show that it can have a negative bias.

Consider a set of $M$ random variables $X = \{X_1, \ldots, X_M\}$. In many problems, one is interested in the maximum expected value of the variables in such a set:

$$\max_i E\{X_i\} \ . \tag{3}$$

Without knowledge of the functional form and parameters of the underlying distributions of the variables in $X$, it is impossible to determine (3) exactly. Most often, this value is approximated by constructing approximations for $E\{X_i\}$ for all $i$. Let $S = \bigcup_{i=1}^{M} S_i$ denote a set of samples, where $S_i$ is the subset containing samples for the variable $X_i$. We assume that the samples in $S_i$ are independent and identically distributed (iid). Unbiased estimates for the expected values can be obtained by computing the sample average for each variable: $E\{X_i\} = E\{\mu_i\} \approx \mu_i(S) \overset{\text{def}}{=} \frac{1}{|S_i|} \sum_{s \in S_i} s$, where $\mu_i$ is an estimator for variable $X_i$. This approximation is unbiased since every sample $s \in S_i$ is an unbiased estimate for the value of $E\{X_i\}$. The error in the approximation thus consists solely of the variance in the estimator and decreases when we obtain more samples.

We use the following notations: $f_i$ denotes the probability density function (PDF) of the $i^{th}$ variable $X_i$ and $F_i(x) = \int_{-\infty}^{x} f_i(x) \, dx$ is the cumulative distribution function (CDF) of this PDF. Similarly, the PDF and CDF of the $i^{th}$ estimator are denoted $f_i^\mu$ and $F_i^\mu$. The maximum expected value can be expressed in terms of the underlying PDFs as $\max_i E\{X_i\} = \max_i \int_{-\infty}^{\infty} x \, f_i(x) \, dx$.

### 2.1   The Single Estimator

An obvious way to approximate the value in (3) is to use the value of the maximal estimator:

$$\max_i E\{X_i\} = \max_i E\{\mu_i\} \approx \max_i \mu_i(S) \ . \tag{4}$$

Because we contrast this method later with a method that uses two estimators for each variable, we call this method the *single estimator*. Q-learning uses this method to approximate the value of the next state by maximizing over the estimated action values in that state.

The maximal estimator $\max_i \mu_i$ is distributed according to some PDF $f^\mu_{\max}$ that is dependent on the PDFs of the estimators $f^\mu_i$. To determine this PDF, consider the CDF $F^\mu_{\max}(x)$, which gives the probability that the maximum estimate is lower or equal to $x$. This probability is equal to the probability that all the estimates are lower or equal to $x$: $F^\mu_{\max}(x) \stackrel{\text{def}}{=} P(\max_i \mu_i \leq x) = \prod_{i=1}^M P(\mu_i \leq x) \stackrel{\text{def}}{=} \prod_{i=1}^M F^\mu_i(x)$. The value $\max_i \mu_i(S)$ is an unbiased estimate for $E\{\max_j \mu_j\} = \int_{-\infty}^\infty x\, f^\mu_{\max}(x)\, dx$, which can thus be given by

$$E\{\max_j \mu_j\} = \int_{-\infty}^\infty x \frac{d}{dx} \prod_{i=1}^M F^\mu_i(x)\, dx \;=\; \sum_j^M \int_{-\infty}^\infty x\, f^\mu_j(s) \prod_{i\neq j}^M F^\mu_i(x)\, dx \quad. \tag{5}$$

However, in (3) the order of the max operator and the expectation operator is the other way around. This makes the maximal estimator $\max_i \mu_i(S)$ a biased estimate for $\max_i E\{X_i\}$. This result has been proven in previous work [16]. A generalization of this proof is included in the supplementary material accompanying this paper.

## 2.2 The Double Estimator

The overestimation that results from the single estimator approach can have a large negative impact on algorithms that use this method, such as Q-learning. Therefore, we look at an alternative method to approximate $\max_i E\{X_i\}$. We refer to this method as the *double estimator*, since it uses two sets of estimators: $\mu^A = \{\mu^A_1, \ldots, \mu^A_M\}$ and $\mu^B = \{\mu^B_1, \ldots, \mu^B_M\}$.

Both sets of estimators are updated with a subset of the samples we draw, such that $S = S^A \cup S^B$ and $S^A \cap S^B = \emptyset$ and $\mu^A_i(S) = \frac{1}{|S^A_i|} \sum_{s \in S^A_i} s$ and $\mu^B_i(S) = \frac{1}{|S^B_i|} \sum_{s \in S^B_i} s$. Like the single estimator $\mu_i$, both $\mu^A_i$ and $\mu^B_i$ are unbiased if we assume that the samples are split in a proper manner, for instance randomly, over the two sets of estimators. Let $Max^A(S) \stackrel{\text{def}}{=} \{j \,|\, \mu^A_j(S) = \max_i \mu^A_i(S)\}$ be the set of maximal estimates in $\mu^A(S)$. Since $\mu^B$ is an independent, unbiased set of estimators, we have $E\{\mu^B_j\} = E\{X_j\}$ for all $j$, including all $j \in Max^A$. Let $a^*$ be an estimator that maximizes $\mu^A$: $\mu^A_{a^*}(S) \stackrel{\text{def}}{=} \max_i \mu^A_i(S)$. If there are multiple estimators that maximize $\mu^A$, we can for instance pick one at random. Then we can use $\mu^B_{a^*}$ as an estimate for $\max_i E\{\mu^B_i\}$ and therefore also for $\max_i E\{X_i\}$ and we obtain the approximation

$$\max_i E\{X_i\} = \max_i E\{\mu^B_i\} \approx \mu^B_{a^*} \quad. \tag{6}$$

As we gain more samples the variance of the estimators decreases. In the limit, $\mu^A_i(S) = \mu^B_i(S) = E\{X_i\}$ for all $i$ and the approximation in (6) converges to the correct result.

Assume that the underlying PDFs are continuous. The probability $P(j = a^*)$ for any $j$ is then equal to the probability that all $i \neq j$ give lower estimates. Thus $\mu^A_j(S) = x$ is maximal for some value $x$ with probability $\prod_{i \neq j}^M P(\mu^A_i < x)$. Integrating out $x$ gives $P(j = a^*) = \int_{-\infty}^\infty P(\mu^A_j = x) \prod_{i \neq j}^M P(\mu^A_i < x)\, dx \stackrel{\text{def}}{=} \int_{-\infty}^\infty f^A_j(x) \prod_{i \neq j}^M F^A_i(x)\, dx$, where $f^A_i$ and $F^A_i$ are the PDF and CDF of $\mu^A_i$. The expected value of the approximation by the double estimator can thus be given by

$$\sum_j^M P(j = a^*) E\{\mu^B_j\} = \sum_j^M E\{\mu^B_j\} \int_{-\infty}^\infty f^A_j(x) \prod_{i \neq j}^M F^A_i(x)\, dx \quad. \tag{7}$$

For discrete PDFs the probability that two or more estimators are equal should be taken into account and the integrals should be replaced with sums. These changes are straightforward.

Comparing (7) to (5), we see the difference is that the double estimator uses $E\{\mu^B_j\}$ in place of $x$. The single estimator overestimates, because $x$ is within integral and therefore correlates with the monotonically increasing product $\prod_{i \neq j} F^\mu_i(x)$. The double estimator underestimates because the probabilities $P(j = a^*)$ sum to one and therefore the approximation is a weighted estimate of unbiased expected values, which must be lower or equal to the maximum expected value. In the following lemma, which holds in both the discrete and the continuous case, we prove in general that the estimate $E\{\mu^B_{a^*}\}$ is not an unbiased estimate of $\max_i E\{X_i\}$.

**Lemma 1.** *Let $X = \{X_1, \ldots, X_M\}$ be a set of random variables and let $\mu^A = \{\mu_1^A, \ldots, \mu_M^A\}$ and $\mu^B = \{\mu_1^B, \ldots, \mu_M^B\}$ be two sets of unbiased estimators such that $E\{\mu_i^A\} = E\{\mu_i^B\} = E\{X_i\}$, for all $i$. Let $\mathcal{M} \overset{\text{def}}{=} \{j \mid E\{X_j\} = \max_i E\{X_i\}\}$ be the set of elements that maximize the expected values. Let $a^*$ be an element that maximizes $\mu^A$: $\mu_{a^*}^A = \max_i \mu_i^A$. Then $E\{\mu_{a^*}^B\} = E\{X_{a^*}\} \leq \max_i E\{X_i\}$. Furthermore, the inequality is strict if and only if $P(a^* \notin \mathcal{M}) > 0$.*

*Proof.* Assume $a^* \in \mathcal{M}$. Then $E\{\mu_{a^*}^B\} = E\{X_{a^*}\} \overset{\text{def}}{=} \max_i E\{X_i\}$. Now assume $a^* \notin \mathcal{M}$ and choose $j \in \mathcal{M}$. Then $E\{\mu_{a^*}^B\} = E\{X_{a^*}\} < E\{X_j\} \overset{\text{def}}{=} \max_i E\{X_i\}$. These two possibilities are mutually exclusive, so the combined expectation can be expressed as

$$
\begin{aligned}
E\{\mu_{a^*}^B\} &= P(a^* \in \mathcal{M})E\{\mu_{a^*}^B | a^* \in \mathcal{M}\} + P(a^* \notin \mathcal{M})E\{\mu_{a^*}^B | a^* \notin \mathcal{M}\} \\
&= P(a^* \in \mathcal{M}) \max_i E\{X_i\} + P(a^* \notin \mathcal{M})E\{\mu_{a^*}^B | a^* \notin \mathcal{M}\} \\
&\leq P(a^* \in \mathcal{M}) \max_i E\{X_i\} + P(a^* \notin \mathcal{M}) \max_i E\{X_i\} \qquad = \max_i E\{X_i\} \;,
\end{aligned}
$$

where the inequality is strict if and only if $P(a^* \notin \mathcal{M}) > 0$. This happens when the variables have different expected values, but their distributions overlap. In contrast with the single estimator, the double estimator is unbiased when the variables are iid, since then all expected values are equal and $P(a^* \in \mathcal{M}) = 1$. $\qquad\qquad\square$

## 3   Double Q-learning

We can interpret Q-learning as using the single estimator to estimate the value of the next state: $\max_a Q_t(s_{t+1}, a)$ is an estimate for $E\{\max_a Q_t(s_{t+1}, a)\}$, which in turn approximates $\max_a E\{Q_t(s_{t+1}, a)\}$. The expectation should be understood as averaging over all possible runs of the same experiment and not—as it is often used in a reinforcement learning context—as the expectation over the next state, which we will encounter in the next subsection as $E\{\cdot|P_t\}$. Therefore, $\max_a Q_t(s_{t+1}, a)$ is an unbiased sample, drawn from an iid distribution with mean $E\{\max_a Q_t(s_{t+1}, a)\}$. In the next section we show empirically that because of this Q-learning can indeed suffer from large overestimations. In this section we present an algorithm to avoid these overestimation issues. The algorithm is called Double Q-learning and is shown in Algorithm 1.

Double Q-learning stores two Q functions: $Q^A$ and $Q^B$. Each Q function is updated with a value from the other Q function for the next state. The action $a^*$ in line 6 is the maximal valued action in state $s'$, according to the value function $Q^A$. However, instead of using the value $Q^A(s', a^*) = \max_a Q^A(s', a)$ to update $Q^A$, as Q-learning would do, we use the value $Q^B(s', a^*)$. Since $Q^B$ was updated on the same problem, but with a different set of experience samples, this can be considered an unbiased estimate for the value of this action. A similar update is used for $Q^B$, using $b^*$ and $Q^A$. It is important that both Q functions learn from separate sets of experiences, but to select an action to perform one can use both value functions. Therefore, this algorithm is not less data-efficient than Q-learning. In our experiments, we calculated the average of the two Q values for each action and then performed $\epsilon$-greedy exploration with the resulting average Q values.

Double Q-learning is not a full solution to the problem of finding the maximum of the expected values of the actions. Similar to the double estimator in Section 2, action $a^*$ may not be the action that maximizes the expected Q function $\max_a E\{Q^A(s', a)\}$. In general $E\{Q^B(s', a^*)\} \leq \max_a E\{Q^A(s', a^*)\}$, and underestimations of the action values can occur.

### 3.1   Convergence in the Limit

In this subsection we show that in the limit Double Q-learning converges to the optimal policy. Intuitively, this is what one would expect: Q-learning is based on the single estimator and Double Q-learning is based on the double estimator and in Section 2 we argued that the estimates by the single and double estimator both converge to the same answer in the limit. However, this argument does not transfer immediately to bootstrapping action values, so we prove this result making use of the following lemma which was also used to prove convergence of Sarsa [20].

**Algorithm 1** Double Q-learning

1: Initialize $Q^A, Q^B, s$
2: **repeat**
3:     Choose $a$, based on $Q^A(s, \cdot)$ and $Q^B(s, \cdot)$, observe $r, s'$
4:     Choose (e.g. random) either UPDATE(A) or UPDATE(B)
5:     **if** UPDATE(A) **then**
6:         Define $a^* = \arg\max_a Q^A(s', a)$
7:         $Q^A(s, a) \leftarrow Q^A(s, a) + \alpha(s, a)\left(r + \gamma Q^B(s', a^*) - Q^A(s, a)\right)$
8:     **else if** UPDATE(B) **then**
9:         Define $b^* = \arg\max_a Q^B(s', a)$
10:      $Q^B(s, a) \leftarrow Q^B(s, a) + \alpha(s, a)(r + \gamma Q^A(s', b^*) - Q^B(s, a))$
11:     **end if**
12:     $s \leftarrow s'$
13: **until** end

**Lemma 2.** *Consider a stochastic process $(\zeta_t, \Delta_t, F_t)$, $t \geq 0$, where $\zeta_t, \Delta_t, F_t : X \to \mathbb{R}$ satisfy the equations:*

$$\Delta_{t+1}(x_t) = (1 - \zeta_t(x_t))\Delta_t(x_t) + \zeta_t(x_t)F_t(x_t) \ , \tag{8}$$

*where $x_t \in X$ and $t = 0, 1, 2, \ldots$. Let $P_t$ be a sequence of increasing $\sigma$-fields such that $\zeta_0$ and $\Delta_0$ are $P_0$-measurable and $\zeta_t, \Delta_t$ and $F_{t-1}$ are $P_t$-measurable, $t = 1, 2, \ldots$. Assume that the following hold: 1) The set X is finite. 2) $\zeta_t(x_t) \in [0, 1]$, $\sum_t \zeta_t(x_t) = \infty$, $\sum_t(\zeta_t(x_t))^2 < \infty$ w.p.1 and $\forall x \neq x_t : \zeta_t(x) = 0$. 3) $||E\{F_t|P_t\}|| \leq \kappa||\Delta_t|| + c_t$, where $\kappa \in [0, 1)$ and $c_t$ converges to zero w.p. 1. 4) $Var\{F_t(x_t)|P_t\} \leq K(1 + \kappa||\Delta_t||)^2$, where $K$ is some constant. Here $|| \cdot ||$ denotes a maximum norm. Then $\Delta_t$ converges to zero with probability one.*

We use this lemma to prove convergence of Double Q-learning under similar conditions as Q-learning. Our theorem is as follows:

**Theorem 1.** *Assume the conditions below are fulfilled. Then, in a given ergodic MDP, both $Q^A$ and $Q^B$ as updated by Double Q-learning as described in Algorithm 1 will converge to the optimal value function $Q^*$ as given in the Bellman optimality equation (2) with probability one if an infinite number of experiences in the form of rewards and state transitions for each state action pair are given by a proper learning policy. The additional conditions are: 1) The MDP is finite, i.e. $|S \times A| < \infty$. 2) $\gamma \in [0, 1)$. 3) The Q values are stored in a lookup table. 4) Both $Q^A$ and $Q^B$ receive an infinite number of updates. 5) $\alpha_t(s, a) \in [0, 1]$, $\sum_t \alpha_t(s, a) = \infty$, $\sum_t(\alpha_t(s, a))^2 < \infty$ w.p.1, and $\forall(s, a) \neq (s_t, a_t) : \alpha_t(s, a) = 0$. 6) $\forall s, a, s' : Var\{R_{sa}^{s'}\} < \infty$.*

A 'proper' learning policy ensures that each state action pair is visited an infinite number of times. For instance, in a communicating MDP proper policies include a random policy.

*Sketch of the proof.* We sketch how to apply Lemma 2 to prove Theorem 1 without going into full technical detail. Because of the symmetry in the updates on the functions $Q^A$ and $Q^B$ it suffices to show convergence for either of these. We will apply Lemma 2 with $P_t = \{Q_0^A, Q_0^B, s_0, a_0, \alpha_0, r_1, s_1, \ldots, s_t, a_t\}$, $X = S \times A$, $\Delta_t = Q_t^A - Q^*$, $\zeta = \alpha$ and $F_t(s_t, a_t) = r_t + \gamma Q_t^B(s_{t+1}, a^*) - Q_t^*(s_t, a_t)$, where $a^* = \arg\max_a Q^A(s_{t+1}, a)$. It is straightforward to show the first two conditions of the lemma hold. The fourth condition of the lemma holds as a consequence of the boundedness condition on the variance of the rewards in the theorem.

This leaves to show that the third condition on the expected contraction of $F_t$ holds. We can write

$$F_t(s_t, a_t) = F_t^Q(s_t, a_t) + \gamma\left(Q_t^B(s_{t+1}, a^*) - Q_t^A(s_{t+1}, a^*)\right) \ ,$$

where $F_t^Q = r_t + \gamma Q_t^A(s_{t+1}, a^*) - Q_t^*(s_t, a_t)$ is the value of $F_t$ if normal Q-learning would be under consideration. It is well-known that $E\{F_t^Q|P_t\} \leq \gamma||\Delta_t||$, so to apply the lemma we identify $c_t = \gamma Q_t^B(s_{t+1}, a^*) - \gamma Q_t^A(s_{t+1}, a^*)$ and it suffices to show that $\Delta_t^{BA} = Q_t^B - Q_t^A$ converges to zero. Depending on whether $Q^B$ or $Q^A$ is updated, the update of $\Delta_t^{BA}$ at time $t$ is either

$$\Delta_{t+1}^{BA}(s_t, a_t) = \Delta_t^{BA}(s_t, a_t) + \alpha_t(s_t, a_t)F_t^B(s_t, a_t) \ , \text{or}$$
$$\Delta_{t+1}^{BA}(s_t, a_t) = \Delta_t^{BA}(s_t, a_t) - \alpha_t(s_t, a_t)F_t^A(s_t, a_t) \ ,$$

where $F_t^A(s_t, a_t) = r_t + \gamma Q_t^B(s_{t+1}, a^*) - Q_t^A(s_t, a_t)$ and $F_t^B(s_t, a_t) = r_t + \gamma Q_t^A(s_{t+1}, b^*) - Q_t^B(s_t, a_t)$. We define $\zeta_t^{BA} = \frac{1}{2}\alpha_t$. Then

$$E\{\Delta_{t+1}^{BA}(s_t, a_t)|P_t\} = \Delta_t^{BA}(s_t, a_t) + E\{\alpha_t(s_t, a_t)F_t^B(s_t, a_t) - \alpha_t(s_t, a_t)F_t^A(s_t, a_t)|P_t\}$$
$$= (1 - \zeta_t^{BA}(s_t, a_t))\Delta_t^{BA}(s_t, a_t) + \zeta_t^{BA}(s_t, a_t)E\{F_t^{BA}(s_t, a_t)|P_t\} \ ,$$

where $E\{F_t^{BA}(s_t, a_t)|P_t\} = \gamma E\left\{Q_t^A(s_{t+1}, b^*) - Q_t^B(s_{t+1}, a^*)|P_t\right\}$. For this step it is important that the selection whether to update $Q^A$ or $Q^B$ is independent on the sample (e.g. random).

Assume $E\{Q_t^A(s_{t+1}, b^*)|P_t\} \geq E\{Q_t^B(s_{t+1}, a^*)|P_t\}$. By definition of $a^*$ as given in line 6 of Algorithm 1 we have $Q_t^A(s_{t+1}, a^*) = \max_a Q_t^A(s_{t+1}, a) \geq Q_t^A(s_{t+1}, b^*)$ and therefore

$$\left|E\{F_t^{BA}(s_t, a_t)|P_t\}\right| = \gamma E\left\{Q_t^A(s_{t+1}, b^*) - Q_t^B(s_{t+1}, a^*)|P_t\right\}$$
$$\leq \gamma E\left\{Q_t^A(s_{t+1}, a^*) - Q_t^B(s_{t+1}, a^*)|P_t\right\} \leq \gamma \left\|\Delta_t^{BA}\right\| \ .$$

Now assume $E\{Q_t^B(s_{t+1}, a^*)|P_t\} > E\{Q_t^A(s_{t+1}, b^*)|P_t\}$ and note that by definition of $b^*$ we have $Q_t^B(s_{t+1}, b^*) \geq Q_t^B(s_{t+1}, a^*)$. Then

$$\left|E\{F_t^{BA}(s_t, a_t)|P_t\}\right| = \gamma E\left\{Q_t^B(s_{t+1}, a^*) - Q_t^A(s_{t+1}, b^*)|P_t\right\}$$
$$\leq \gamma E\left\{Q_t^B(s_{t+1}, b^*) - Q_t^A(s_{t+1}, b^*)|P_t\right\} \leq \gamma \left\|\Delta_t^{BA}\right\| \ .$$

Clearly, one of the two assumptions must hold at each time step and in both cases we obtain the desired result that $|E\{F_t^{BA}|P_t\}| \leq \gamma\|\Delta_t^{BA}\|$. Applying the lemma yields convergence of $\Delta_t^{BA}$ to zero, which in turn ensures that the original process also converges in the limit. $\qquad\square$

## 4 Experiments

This section contains results on two problems, as an illustration of the bias of Q-learning and as a first practical comparison with Double Q-learning. The settings are simple to allow an easy interpretation of what is happening. Double Q-learning scales to larger problems and continuous spaces in the same way as Q-learning, so our focus here is explicitly on the bias of the algorithms.

The settings are the gambling game of roulette and a small grid world. There is considerable randomness in the rewards, and as a result we will see that indeed Q-learning performs poorly. The discount factor was 0.95 in all experiments. We conducted two experiments on each problem. The learning rate was either linear: $\alpha_t(s, a) = 1/n_t(s, a)$, or polynomial $\alpha_t(s, a) = 1/n_t(s, a)^{0.8}$. For Double Q-learning $n_t(s, a) = n_t^A(s, a)$ if $Q^A$ is updated and $n_t(s, a) = n_t^B(s, a)$ if $Q^B$ is updated, where $n_t^A$ and $n_t^B$ store the number of updates for each action for the corresponding value function. The polynomial learning rate was shown in previous work to be better in theory and in practice [14].

### 4.1 Roulette

In roulette, a player chooses between 170 betting actions, including betting on a number, on either of the colors black or red, and so on. The payoff for each of these bets is chosen such that almost all bets have an expected payout of $\frac{1}{38}\$36 = \$0.947$ per dollar, resulting in an expected loss of -\$0.053 per play if we assume the player bets \$1 every time.[1] We assume all betting actions transition back to the same state and there is one action that stops playing, yielding \$0. We ignore the available funds of the player as a factor and assume he bets \$1 each turn.

Figure 1 shows the mean action values over all actions, as found by Q-learning and Double Q-learning. Each trial consisted of a synchronous update of all 171 actions. After 100,000 trials, Q-learning with a linear learning rate values all betting actions at more than \$20 and there is little progress. With polynomial learning rates the performance improves, but Double Q-learning converges much more quickly. The average estimates of Q-learning are not poor because of a few poorly estimated outliers. After 100,000 trials Q-learning valued all non-terminating actions between \$22.63 and \$22.67 for linear learning rates and between \$9.58 to \$9.64 for polynomial rates. In this setting Double Q-learning does not suffer from significant underestimations.

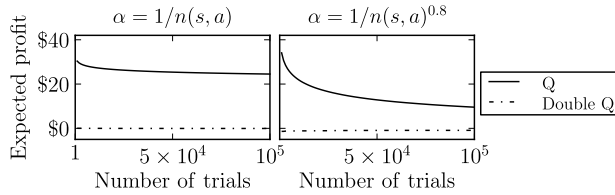

Figure 1: The average action values according to Q-learning and Double Q-learning when playing roulette. The 'walk-away' action is worth $0. Averaged over 10 experiments.

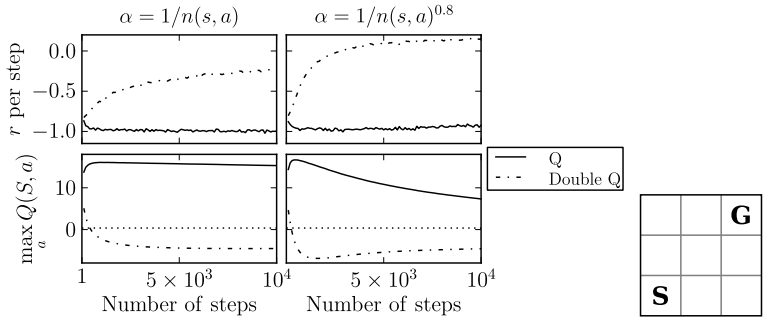

Figure 2: Results in the grid world for Q-learning and Double Q-learning. The first row shows average rewards per time step. The second row shows the maximal action value in the starting state S. Averaged over 10,000 experiments.

## 4.2 Grid World

Consider the small grid world MDP as show in Figure 2. Each state has 4 actions, corresponding to the directions the agent can go. The starting state is in the lower left position and the goal state is in the upper right. Each time the agent selects an action that walks off the grid, the agent stays in the same state. Each non-terminating step, the agent receives a random reward of $-12$ or $+10$ with equal probability. In the goal state every action yields $+5$ and ends an episode. The optimal policy ends an episode after five actions, so the optimal average reward per step is $+0.2$. The exploration was $\epsilon$-greedy with $\epsilon(s) = 1/\sqrt{n(s)}$ where $n(s)$ is the number of times state $s$ has been visited, assuring infinite exploration in the limit which is a theoretical requirement for the convergence of both Q-learning and Double Q-learning. Such an $\epsilon$-greedy setting is beneficial for Q-learning, since this implies that actions with large overestimations are selected more often than realistically valued actions. This can reduce the overestimation.

Figure 2 shows the average rewards in the first row and the maximum action value in the starting state in the second row. Double Q-learning performs much better in terms of its average rewards, but this does not imply that the estimations of the action values are accurate. The optimal value of the maximally valued action in the starting state is $5\gamma^4 - \sum_{k=0}^{3} \gamma^k \approx 0.36$, which is depicted in the second row of Figure 2 with a horizontal dotted line. We see Double Q-learning does not get much closer to this value in $10,000$ learning steps than Q-learning. However, even if the error of the action values is comparable, the policies found by Double Q-learning are clearly much better.

## 5 Discussion

We note an important difference between the well known heuristic exploration technique of optimism in the face of uncertainty [21, 22] and the overestimation bias. Optimism about uncertain events can be beneficial, but Q-learning can overestimate actions that have been tried often and the estimations can be higher than any realistic optimistic estimate. For instance, in roulette our initial action value estimate of $0 can be considered optimistic, since no action has an actual expected value higher than this. However, even after trying 100,000 actions Q-learning on average estimated each gambling action to be worth almost $10. In contrast, although Double Q-learning can underestimate

the values of some actions, it is easy to set the initial action values high enough to ensure optimism for actions that have experienced limited updates. Therefore, the use of the technique of optimism in the face of uncertainty can be thought of as an orthogonal concept to the over- and underestimation that is the topic of this paper.

The analysis in this paper is not only applicable to Q-learning. For instance, in a recent paper on multi-armed bandit problems, methods were proposed to exploit structure in the form of the presence of clusters of correlated arms in order to speed up convergence and reduce total regret [23]. The value of such a cluster in itself is an estimation task and the proposed methods included taking the mean value, which would result in an underestimation of the actual value, and taking the maximum value, which is a case of the single estimator and results in an overestimation. It would be interesting to see how the double estimator approach fares in such a setting.

Although the settings in our experiments used stochastic rewards, our analysis is not limited to MDPs with stochastic reward functions. When the rewards are deterministic but the state transitions are stochastic, the same pattern of overestimations due to this noise can occur and the same conclusions continue to hold.

## 6   Conclusion

We have presented a new algorithm called Double Q-learning that uses a double estimator approach to determine the value of the next state. To our knowledge, this is the first off-policy value based reinforcement learning algorithm that does not have a positive bias in estimating the action values in stochastic environments. According to our analysis, Double Q-learning sometimes underestimates the action values, but does not suffer from the overestimation bias that Q-learning does. In a roulette game and a maze problem, Double Q-learning was shown to reach good performance levels much more quickly.

**Future work**   Interesting future work would include research to obtain more insight into the merits of the Double Q-learning algorithm. For instance, some preliminary experiments in the grid world showed that Q-learning performs even worse with higher discount factors, but Double Q-learning is virtually unaffected. Additionally, the fact that we can construct positively biased and negatively biased off-policy algorithms raises the question whether it is also possible to construct an unbiased off-policy reinforcement-learning algorithm, without the high variance of unbiased on-policy Monte-Carlo methods [24]. Possibly, this can be done by estimating the size of the overestimation and deducting this from the estimate. Unfortunately, the size of the overestimation is dependent on the number of actions and the unknown distributions of the rewards and transitions, making this a non-trivial extension.

More analysis on the performance of Q-learning and related algorithms such as Fitted Q-iteration [12] and Delayed Q-learning [10] is desirable. For instance, Delayed Q-learning can suffer from similar overestimations, although it does have polynomial convergence guarantees. This is similar to the polynomial learning rates: although performance is improved from an exponential to a polynomial rate [14], the algorithm still suffers from the inherent overestimation bias due to the single estimator approach. Furthermore, it would be interesting to see how Fitted Double Q-iteration, Delayed Double Q-learning and other extensions of Q-learning perform in practice when they are applied to Double Q-learning.

### Acknowledgments

The authors wish to thank Marco Wiering and Gerard Vreeswijk for helpful comments. This research was made possible thanks to grant 612.066.514 of the dutch organization for scientific research (Nederlandse Organisatie voor Wetenschappelijk Onderzoek, NWO).

## Footnotes

[1]Only the so called 'top line' which pays \$6 per dollar when 00, 0, 1, 2 or 3 is hit has a slightly lower expected value of -\$0.079 per dollar.

## References

[1] C. J. C. H. Watkins. *Learning from Delayed Rewards*. PhD thesis, King's College, Cambridge, England, 1989.

[2] C. J. C. H. Watkins and P. Dayan. Q-learning. *Machine Learning*, 8:279–292, 1992.

[3] R. Bellman. *Dynamic Programming*. Princeton University Press, 1957.

[4] T. Jaakkola, M. I. Jordan, and S. P. Singh. On the convergence of stochastic iterative dynamic programming algorithms. *Neural Computation*, 6:1185–1201, 1994.

[5] J. N. Tsitsiklis. Asynchronous stochastic approximation and Q-learning. *Machine Learning*, 16:185–202, 1994.

[6] M. L. Littman and C. Szepesvári. A generalized reinforcement-learning model: Convergence and applications. In L. Saitta, editor, *Proceedings of the 13th International Conference on Machine Learning (ICML-96)*, pages 310–318, Bari, Italy, 1996. Morgan Kaufmann.

[7] R. H. Crites and A. G. Barto. Improving elevator performance using reinforcement learning. In D. S. Touretzky, M. C. Mozer, and M. E. Hasselmo, editors, *Advances in Neural Information Processing Systems 8*, pages 1017–1023, Cambridge MA, 1996. MIT Press.

[8] W. D. Smart and L. P. Kaelbling. Effective reinforcement learning for mobile robots. In *Proceedings of the 2002 IEEE International Conference on Robotics and Automation (ICRA 2002)*, pages 3404–3410, Washington, DC, USA, 2002.

[9] M. A. Wiering and H. P. van Hasselt. Ensemble algorithms in reinforcement learning. *IEEE Transactions on Systems, Man, and Cybernetics, Part B*, 38(4):930–936, 2008.

[10] A. L. Strehl, L. Li, E. Wiewiora, J. Langford, and M. L. Littman. PAC model-free reinforcement learning. In *Proceedings of the 23rd international conference on Machine learning*, pages 881–888. ACM, 2006.

[11] M. J. Kearns and S. P. Singh. Finite-sample convergence rates for Q-learning and indirect algorithms. In *Neural Information Processing Systems 12*, pages 996–1002. MIT Press, 1999.

[12] D. Ernst, P. Geurts, and L. Wehenkel. Tree-based batch mode reinforcement learning. *Journal of Machine Learning Research*, 6(1):503–556, 2005.

[13] C. Szepesvári. The asymptotic convergence-rate of Q-learning. In *NIPS '97: Proceedings of the 1997 conference on Advances in neural information processing systems 10*, pages 1064–1070, Cambridge, MA, USA, 1998. MIT Press.

[14] E. Even-Dar and Y. Mansour. Learning rates for Q-learning. *Journal of Machine Learning Research*, 5:1–25, 2003.

[15] E. Van den Steen. Rational overoptimism (and other biases). *American Economic Review*, 94(4):1141–1151, September 2004.

[16] J. E. Smith and R. L. Winkler. The optimizer's curse: Skepticism and postdecision surprise in decision analysis. *Management Science*, 52(3):311–322, 2006.

[17] E. Capen, R. Clapp, and T. Campbell. Bidding in high risk situations. *Journal of Petroleum Technology*, 23:641–653, 1971.

[18] R. H. Thaler. Anomalies: The winner's curse. *Journal of Economic Perspectives*, 2(1):191–202, Winter 1988.

[19] J. L. W. V. Jensen. Sur les fonctions convexes et les inégalités entre les valeurs moyennes. *Journal Acta Mathematica*, 30(1):175–193, 1906.

[20] S. P. Singh, T. Jaakkola, M. L. Littman, and C. Szepesvári. Convergence results for single-step on-policy reinforcement-learning algorithms. *Machine Learning*, 38(3):287–308, 2000.

[21] L. P. Kaelbling, M. L. Littman, and A. W. Moore. Reinforcement learning: A survey. *Journal of Artificial Intelligence Research*, 4:237–285, 1996.

[22] R. S. Sutton and A. G. Barto. *Reinforcement Learning: An Introduction*. The MIT press, Cambridge MA, 1998.

[23] S. Pandey, D. Chakrabarti, and D. Agarwal. Multi-armed bandit problems with dependent arms. In *Proceedings of the 24th international conference on Machine learning*, pages 721–728. ACM, 2007.

[24] W. K. Hastings. Monte Carlo sampling methods using Markov chains and their applications. *Biometrika*, pages 97–109, 1970.

